# Discrete profile alignment via constrained information bottleneck

**Sean O'Rourke**[*]
seano@cs.ucsd.edu

**Gal Chechik**[†]
gal@stanford.edu

**Robin Friedman**[*]
rcfriedm@ucsd.edu

**Eleazar Eskin**[*]
eeskin@cs.ucsd.edu

## Abstract

Amino acid profiles, which capture position-specific mutation probabilities, are a richer encoding of biological sequences than the individual sequences themselves. However, profile comparisons are much more computationally expensive than discrete symbol comparisons, making profiles impractical for many large datasets. Furthermore, because they are such a rich representation, profiles can be difficult to visualize. To overcome these problems, we propose a discretization for profiles using an expanded alphabet representing not just individual amino acids, but common profiles. By using an extension of information bottleneck (IB) incorporating constraints and priors on the class distributions, we find an informationally optimal alphabet. This discretization yields a concise, informative textual representation for profile sequences. Also alignments between these sequences, while nearly as accurate as the full profile-profile alignments, can be computed almost as quickly as those between individual or consensus sequences. A full pairwise alignment of SwissProt would take years using profiles, but less than 3 days using a discrete IB encoding, illustrating how discrete encoding can expand the range of sequence problems to which profile information can be applied.

## 1 Introduction

One of the most powerful techniques in protein analysis is the comparison of a target amino acid sequence with phylogenetically related or homologous proteins. Such comparisons give insight into which portions of the protein are important by revealing the parts that were conserved through natural selection. While mutations in non-functional regions may be harmless, mutations in functional regions are often lethal. For this reason, functional regions of a protein tend to be conserved between organisms while non-functional regions diverge.

---

[*]Department of Computer Science and Engineering, University of California San Diego
[†]Department of Computer Science, Stanford University

Many of the state-of-the-art protein analysis techniques incorporate homologous sequences by representing a set of homologous sequences as a probabilistic *profile*, a sequence of the marginal distributions of amino acids at each position in the sequence. For example, Yona et al.[10] uses profiles to align distant homologues from the SCOP database[3]; the resulting alignments are similar to results from structural alignments, and tend to reflect both secondary and tertiary protein structure. The PHD algorithm[5] uses profiles purely for structure prediction. PSI–BLAST[6] uses them to refine database searches.

Although profiles provide a lot of information about the sequence, the use of profiles comes at a steep price. While extremely efficient string algorithms exist for aligning protein sequences (Smith-Waterman[8]) and performing database queries (BLAST[6]), these algorithms operate on strings and are not immediately applicable to profile alignment or profile database queries. While profile-based methods can be substantially more accurate than sequence-based ones, they can require at least an order of magnitude more computation time, since substitution penalties must be calculated by computing distances between probability distributions. This makes profiles impractical for use with large bioinformatics databases like SwissProt, which recently passed 150,000 sequences. Another drawback of profile as compared to string representations is that it is much more difficult to visually interpret a sequence of 20 dimensional vectors than a sequence of letters.

Discretizing the profiles addresses both of these problems. First, once a profile is represented using a discrete alphabet, alignment and database search can be performed using the efficient string algorithms developed for sequences. For example, when aligning sequences of 1000 elements, runtime decreases from 20 seconds for profiles to 2 for discrete sequences. Second, by representing each class as a letter, discretized profiles can be presented in plain text like the original or consensus sequences, while conveying more information about the underlying profiles. This makes them more accurate than consensus sequences, and more dense than sequence logos (see figure 1). To make this representation intuitive, we want the discretization not only to minimize information loss, but also to reflect biologically meaningful categories by forming a superset of the standard 20-character amino acid alphabet. For example, we use "A" and "a" for strongly- and weakly-conserved Alanine. This formulation demands two types of constraints: similarities of the centroids to predefined values, and specific structural similarities between strongly- and weakly-conserved variants. We show below how these constraints can be added to the original IB formalism.

In this paper, we present a new discrete representation of proteins that takes into account information from homologues. The main idea behind our approach is to compress the space of probabilistic profiles in a data-dependent manner by clustering the actual profiles and representing them by a small alphabet of distributions. Since this discretization removes some of the information carried by the full profiles, we cluster the distribution in a way that is directly targeted at minimizing the information loss. This is achieved using a variant of Information Bottleneck (IB)[9], a distributional clustering approach for informationally optimal discretization.

We apply our algorithm to a subset of MEROPS[4], a database of peptidases organized structurally by family and clan, and analyze the results in terms of both information loss and alignment quality. We show that multivariate IB in particular preserves much of the information in the original profiles using a small number of classes. Furthermore, optimal alignments for profile sequences encoded with these classes are much closer to the original profile-profile alignments than are alignments between the seed proteins. IB discretization is therefore an attractive way to gain some of the additional sensitivity of profiles with less computational cost.

```
0.0   0.0   0.0   0.09   0.34   0.23   0.12   0.0   0.0   0.0
0.0   0.0   0.0   0.04   0.01   0.01   0.03   0.0   0.0   0.0
0.0   0.0   1.0   0.01   0.05   0.14   0.09   0.0   1.0   0.0
0.0   0.0   0.0   0.38   0.04   0.00   0.04   0.0   0.0   0.0
0.0   0.0   0.0   0.06   0.00   0.08   0.04   0.0   0.0   1.0
0.0   0.0   0.0   0.00   0.06   0.01   0.03   1.0   0.0   0.0
0.0   0.0   0.0   0.02   0.00   0.04   0.00   0.0   0.0   0.0
0.0   0.0   0.0   0.00   0.00   0.03   0.00   0.0   0.0   0.0
0.0   0.0   0.0   0.04   0.01   0.01   0.00   0.0   0.0   0.0
0.0   0.0   0.0   0.01   0.01   0.00   0.09   0.0   0.0   0.0
0.0   0.0   0.0   0.00   0.00   0.03   0.00   0.0   0.0   0.0
0.5   1.0   0.0   0.05   0.05   0.01   0.01   0.0   0.0   0.0
0.0   0.0   0.0   0.02   0.00   0.23   0.00   0.0   0.0   0.0
0.0   0.0   0.0   0.04   0.05   0.00   0.00   0.0   0.0   0.0
0.0   0.0   0.0   0.04   0.01   0.00   0.00   0.0   0.0   0.0
0.5   0.0   0.0   0.16   0.10   0.06   0.29   0.0   0.0   0.0
0.0   0.0   0.0   0.02   0.10   0.05   0.20   0.0   0.0   0.0
0.0   0.0   0.0   0.00   0.14   0.03   0.04   0.0   0.0   0.0
0.0   0.0   0.0   0.00   0.00   0.00   0.00   0.0   0.0   0.0
0.0   0.0   0.0   0.01   0.00   0.04   0.04   0.0   0.0   0.0
```

(a)

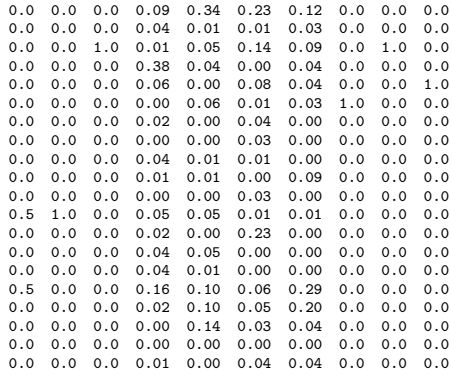

(b)

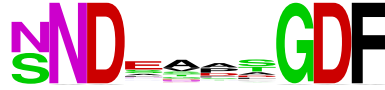

P00790 Seq.:      ---EAPT---
Consensus Seq.:   NNDEAASGDF
IB Seq.:          NNDeaptGDF

(c)

Figure 1: (a) Profile, (b) sequence logo[2], and (c) textual representations for part of an alignment of Pepsin A precursor P00790, showing IB's concision compared to profiles and logos, and its precision compared to single sequences.

## 2 Information Bottleneck

Information Bottleneck [9] is an information theoretic approach for distributional clustering. Given a joint distribution $p(X, Y)$ of two random variables $X$ and $Y$, the goal is to obtain a compressed representation $C$ of $X$, while preserving the information about $Y$. The two goals of compression and information preservation are quantified by the same measure of mutual information $I(X; Y) = \sum_{x,y} p(x, y) \log \frac{p(x,y)}{p(x)p(y)}$ and the problem is therefore defined as the constrained optimization problem $\min_{p(c|x):I(C;Y)>K} I(C; X)$ where $K$ is a constraint on the level of information preserved about $Y$, and the problem should also obey the constraints $p(y|c) = \sum_x p(y|x)p(x|c)$ and $p(y) = \sum_x p(y|x)p(x)$. This constrained optimization can be reformulated using Lagrange multipliers, and turned into a tradeoff optimization function with Lagrange multiplier $\beta$:

$$\min_{p(c|x)} \mathcal{L} \stackrel{\text{def}}{=} I(C; X) - \beta I(C; Y) \tag{1}$$

As an unsupervised learning technique, IB aims to characterize the set of solutions for the complete spectrum of constraint values $K$. This set of solutions is identical to the set of solutions of the tradeoff optimization problem obtained for the spectrum of $\beta$ values.

When $X$ is discrete, its natural compression is fuzzy clustering. In this case, the problem is not convex and cannot be guaranteed to contain a single global minimum. Fortunately, its solutions can be characterized analytically by a set of self consistent equations. These self consistent equations can then be used in an iterative algorithm that is guaranteed to converge to a local minimum. While the optimal solutions of the IB functional are in general soft clusters, in practice, hard cluster solutions are sometimes more easily interpreted. A series of algorithms was developed for hard IB, including an algorithm that can be viewed as a one-step look-ahead sequential version of K-Means [7].

To apply IB to the problem of profiles discretization discussed here, $X$ is a given set of probabilistic profiles obtained from a set of aligned sequences and $Y$ is the set of 20 amino acids.

## 2.1 Constraints on centroids' semantics

The application studied in this paper differs from standard IB applications in that we are interested in obtaining a representation that is both efficient and biologically meaningful. This requires that we add two kinds of constraints on clusters' distributions, discussed below.

First, some clusters' meanings are naturally determined by limiting them to correspond to the common 20-letter alphabet used to describe amino acids. From the point of view of distributions over amino acids, each of these symbols is used today as the delta function distribution which is fully concentrated on a single amino acid. For the goal of finding an efficient representation, we require the centroids to be close to these delta distributions. More generally, we require the centroids to be close to some predefined values $\hat{c}_i$, thus adding constraints to the IB target function of the form $D_{KL}[p(y|\hat{c}_i)||p(y|c_i)] < K_i$ for each constrained centroid. While solving the constrained optimization problem is difficult, the corresponding tradeoff optimization problem can be made very similar to standard IB. With the additional constraints, the IB trade-off optimization problem becomes

$$\min_{p(c|x)} \mathcal{L}' \equiv I(C;X) - \beta I(C;Y) + \beta \sum_{c_i \in C} \beta(c_i) D_{KL}[p(y|\hat{c}_i)||p(y|c_i)] \quad . \qquad (2)$$

We now use the following identity

$$\sum_{x,c} p(x,c) D_{KL}[p(y|x)||p(y|c)]$$

$$= \sum_x p(x) \sum_y p(y|x) \log p(y|x) - \sum_c p(c) \sum_y \log p(y|c) \sum_x p(y|x) p(x|c)$$

$$= -H(Y|X) + H(Y|C) = I(X;Y) - I(Y;C)$$

to rewrite the IB functional of Eq. (1) as

$$\mathcal{L} = I(C;X) + \beta \sum_{c \in C} \sum_{x \in X} p(x,c) D_{KL}[p(y|x)||p(y|c)] - \beta I(X;Y)$$

When $\sum \beta(c_i) \leq 1$ we can similarly rewrite Eq. (2) as

$$\mathcal{L}' = I(C;X) + \beta \sum_{x \in X} p(x) \sum_{c_i \in C} p(c_i|x) D_{KL}[p(y|x)||p(y|c_i)] \qquad (3)$$

$$+ \beta \sum_{c_i \in C} \beta(c_i) D_{KL}[p(y|\hat{c}_i)||p(y|c_i)] - \beta I(X;Y)$$

$$= I(C;X) + \beta \sum_{x' \in X'} p(x') \sum_{c_i \in C} p(c_i|x') D_{KL}[p(y|x')||p(y|c_i)] - \beta I(X;Y)$$

The optimization problem therefore becomes equivalent to the original IB problem, but with a modified set of samples $x \in X'$, containing $X$ plus additional "pseudo-counts" or biases. This is similar to the inclusion of priors in Bayesian estimation. Formulated this way, the biases can be easily incorporated in standard IB algorithms by adding additional pseudo-counts $x'$ with prior probability $p(x') = \beta_i(c)$.

## 2.2 Constraints on relations between centroids

We want our discretization to capture correlations between strongly- and weakly-conserved variants of the same symbol. This can be done with standard IB using

separate classes for the alternatives. However, since the distributions of other amino acids in these two variants are likely to be related, it is preferable to define a single shared prior for both variants, and to learn a model capturing their correlation.

Friedman et al.[1] describe *multivariate information bottleneck* (mIB), an extension of information bottleneck to joint distributions over several correlated input and cluster variables. For profile discretization, we define two compression variables connected as in Friedman's "parallel IB": an amino acid class $C \in \{A, C, \ldots\}$ with an associated prior, and a strength $S \in \{0, 1\}$. Since this model correlates strong and weak variants of each category, it requires fewer priors than simple IB. It also has fewer parameters: a multivariate model with $n_s$ strengths and $n_c$ classes has as many categories as a univariate one with $n_{c'} = n_s n_c$ classes, but has only $n_s + n_c - 2$ free parameters for each $x$, instead of $n_s n_c - 1$.

## 3   Results

To test our method, we apply it to data from MEROPS[4]. Proteins within the same family typically contain high-confidence alignments, those from different families in the same clan less so. For each protein, we generate a profile from alignments obtained from PSI–BLAST with standard parameters, and compute IB classes from a large subset of these profiles using the priors described below. Finally, we encode and align pairs of profiles using the learned classes, comparing the results to those obtained both with the full profiles and with just the original sequences.

For univariate IB, we have used four types of priors reflecting biases on stability, physical properties, and observed substitution frequencies: (1) *Strongly conserved* classes, in which a single symbol is seen with $S\%$ probability. These are the only priors used for multivariate IB. (2) *Weakly conserved* classes, in which a single symbol occurs with $W\%$ probability; $(S-W)\%$ of the remaining probability mass is distributed among symbols with non-negative log-odds of substitution. (3) *Physical trait* classes, in which all symbols with the same hydrophobicity, charge, polarity, or aromaticity occur uniformly $S\%$ of the time. (4) A *uniform* class, in which all symbols occur with their background probabilities.

The choice of $S$ and $W$ depends upon both the data and one's prior notions of "strong" and "weak" conservation. Unbiased IB on a large subset of MEROPS with several different numbers of unbiased categories yielded a mean frequency approaching 0.7 for the most common symbol in the 20 most sharply-distributed classes ($0.59 \pm 0.13$ for $|C| = 52$; $0.66 \pm 0.12$ for $|C| = 80$; $0.70 \pm 0.09$ for $|C| = 100$). Similarly, the next 20 classes have a mean most-likely-symbol frequency around 0.4. These numbers can be seen as lower bounds on $S$ and $W$. We therefore chose $S = 0.8$ and $W = 0.5$, reflecting a bias toward stronger definitions of conservation than those inferred from the data.

### 3.1   Iterative vs. Sequential IB

Slonim[7] compares several IB algorithms, concluding that best hard clustering results are obtained with a sequential method (sIB), in which elements are first assigned to a fixed number of clusters and then individually moved from cluster to cluster while calculating a 1-step lookahead score, until the score converges. While sIB is more efficient than exhaustive bottom-up clustering, it neglects information about the best potential candidates to be assigned to a cluster, yielding slow convergence. Furthermore updates are expensive, since each requires recomputing the class centroids. Therefore instead of sIB, we use iterative IB (iIB) with hard clustering, which only recomputes the centroids after performing all updates. This reduces

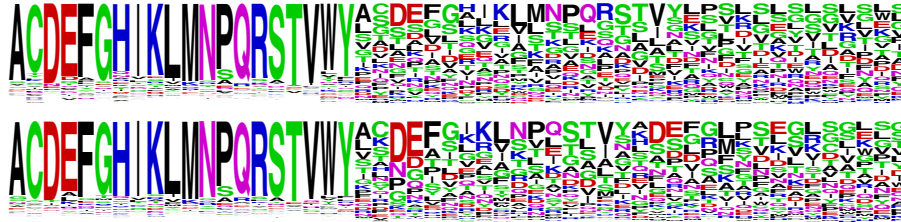

Figure 2: Stretched sequence logos for categories found by iIB (top) and sIB (bottom), ordered by primary symbol and decreasing information.

the convergence time from several hours to around ten minutes.

Since Slonim argues that sIB outperforms soft iIB in part because sIB's discrete steps allow it to escape local optima, we expect hard iIB to have similar behavior. To test this, we applied three complete sIB iterations initialized with categories from multivariate iIB. sIB decreased the loss $\mathcal{L}$ by only about 3 percent (from 0.380 to 0.368), with most of this gain occurring in the first iteration. Also, the resulting categories were mostly conserved up to exchanging labels, suggesting that hard iIB finds categories similar sIB ones (see figure 2).

## 3.2   Information Loss and Alignments

One measure of the quality of the resulting clusters is the amount of information about $Y$ lost through discretization, $I(Y;X) - I(Y;C)$. Figure (3b) shows the effect on information loss of varying the prior weight $w$ with three sets of priors: 20 strongly conserved symbols and one background; these plus 20 weakly conserved symbols; and these plus 10 categories for physical characteristics. As expected, both decreasing the number of categories and increasing the number or weight of priors increases information loss. However, with a fixed number of free categories, information loss is nearly independent of prior strength, suggesting that our priors correspond to actual regularities in the data. Finally, note that despite having fewer free parameters than the univariate models, mIB's achieves comparable performance, suggesting that our decomposition into conserved class and degree of conservation is reasonable.

Since we are ultimately using these classes in alignments, the true cost of discretization is best measured by the amount of change between profile and IB alignments, and the significance of this change. The latter is important because the best path can be very sensitive to small changes in the sequences or scoring matrix; if two radically different alignments have similar scores, neither is clearly "correct". We can represent an alignment as a pair of index-insertion sequences, one for each profile sequence to be aligned (e.g. "1,2,_,_,3,..." versus "1,_,2,_,3,..."). The edit distance between these sequences for two alignments then measures how much they differ. However, even when this distance is large, the difference between two alignments may not be significant if both choices' scores are nearly the same. That is, if the optimal profile alignment's score is only slightly lower than the optimal IB class alignment's score *as computed with the original profiles*, either might be correct.

Figure 4 shows at left both the edit distance and score change per length between profile alignments and those using IB classes, mIB classes, and the original sequences with the BLOSUM62 scoring matrix. To compare the profile and sequence alignments, profiles corresponding to gaps in the original sequences are replaced

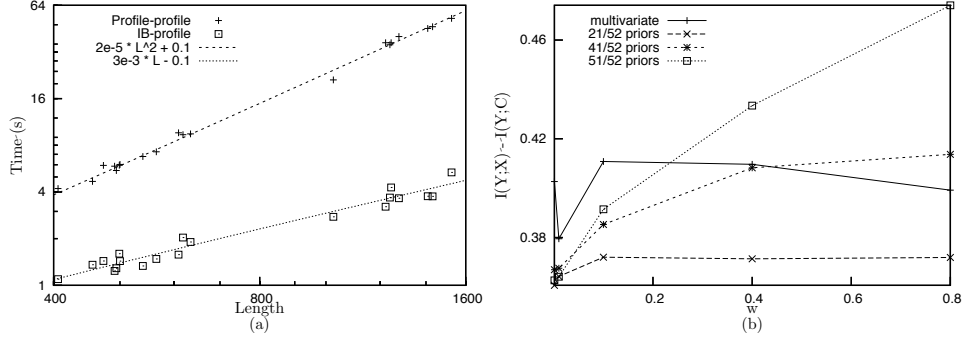

Figure 3: (a) Running times for profile-profile versus IB-profile alignment, showing speedups of 3.5-12.5x for pairwise global alignment. (b)$I(Y;X) - I(Y;C)$ as a function of $w$ for different groups of priors. The information loss for 52 categories without priors is 0.359, for 10, 0.474.

|  | Edit distance | Score change |
|---|---|---|
|  | Same Superfamily | |
| mIB | $0.154 \pm 0.182$ | $0.086 \pm 0.166$ |
| IB | $0.170 \pm 0.189$ | $0.107 \pm 0.198$ |
| BLOSUM | $0.390 \pm 0.065$ | |
|  | Same Clan | |
| mIB | $0.124 \pm 0.209$ | $0.019 \pm 0.029$ |
| IB | $0.147 \pm 0.232$ | $0.022 \pm 0.037$ |
| BLOSUM | $0.360 \pm 0.062$ | |

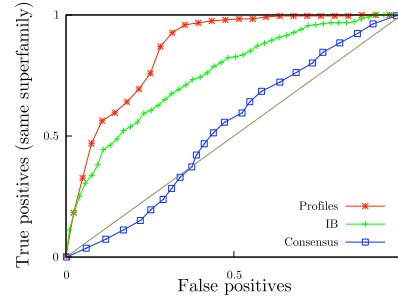

Figure 4: Left: alignment differences for IB models and sequence alignment, within and between superfamilies. Right: ROC curve for same/different superfamily classification by alignment score.

by gaps, and resulting pairs of aligned gaps in the profile-profile alignment are removed. We consider both sequences from the same family and those from other families in the same clan, the former being more similar than the latter, and therefore having better alignments. Assuming the profile-profile alignment is closest to the "true" alignment, iIB alignment significantly outperforms sequence alignment in both cases, with mIB showing a slight additional improvement. At right is the ROC curve for detecting superfamily relationships between profiles from different families based on alignment scores, showing that while IB fares worse than profiles, simple sequences perform essentially at chance.

Finally, figure 3a compares the performance of profile and IB alignment for different sequence lengths. To use a profile alphabet for novel alignments, we must map each input profile to the closest IB class. To be consistent with Yona[10], we use the Jensen-Shannon (JS) distance with mixing coefficient 0.5 rather than the KL distance optimized in creating the categories. Aligning two sequences of lengths $n$ and $m$ requires computing the $|C|(n+m)$ JS-distances between each profile and each category, a significant improvement over the $mn$ distance computations required for profile-profile alignment when $|C| \ll \frac{\min(m,n)}{2}$. Our results show that JS distance computations dominate running time, since IB alignment time scales linearly with the input size, while profile alignment scales quadratically, yielding an order of magnitude improvement for typical 500- to 1000-base-pair sequences.

# 4    Discussion

We have described a discrete approximation to amino acid profiles, based on minimizing information loss, that allows profile information to be used for alignment and search without additional computational cost compared to simple sequence alignment. Alignments of sequences encoded with a modest number of classes correspond to the original profile alignments significantly better than alignments of the original sequences. In addition to minimizing information loss, the classes can be constrained to correspond to the standard amino acid representation, yielding an intuitive, compact textual form for profile information.

Our model is useful in three ways: (1) it makes it possible to apply existing fast discrete algorithms to arbitrary continuous sequences; (2) it models rich conditional distribution structures; and (3) its models can incorporate a variety of class constraints. We can extend our approach in each of these directions. For example, adjacent positions are highly correlated: the average entropy of a single profile is 0.99, versus 1.23 for an adjacent pair. Therefore pairs can be represented more compactly than the cross-product of a single-position alphabet. More generally, we can encode arbitrary conserved regions and still treat them symbolically for alignment and search. Other extensions include incorporating structural information in the input representation; assigning structural significance to the resulting categories; and learning multivariate IB's underlying model's structure.

# References

[1] Nir Friedman, Ori Mosenzon, Noam Slonim, and Naftali Tishby. Multivariate information bottleneck. In *Uncertainty in Artificial Intelligence: Proceedings of the Seventeenth Conference (UAI-2001)*, pages 152–161, San Francisco, CA, 2001. Morgan Kaufmann Publishers.

[2] Crooks GE, Hon G, Chandonia JM, and Brenner SE. WebLogo: a sequence logo generator. *Genome Research*, in press, 2004.

[3] A. G. Murzin, S. E. Brenner, T. Hubbard, and C. Chothia. SCOP: a structural classification of proteins database for the investigation of sequences and structures. *J. Mol. Biol.*, 247:536–40, 1995.

[4] N.D. Rawlings, D.P. Tolle, and A.J. Barrett. MEROPS: the peptidase database. *Nucleic Acids Res*, 32 Database issue:D160–4, 2004.

[5] B. Rost and C. Sander. Prediction of protein secondary structure at better than 70% accuracy. *J. Mol. Bio.*, 232:584–99, 1993.

[6] Altschul SF, Gish W, Miller W, Myers EW, and Lipman DJ. Basic local alignment search tool. *J Mol Biol*, 215(3):403–10, October 1990.

[7] Noam Slonim. *The Information Bottleneck: Theory and Applications.* PhD thesis, Hebrew University, Jerusalem, Israel, 2002.

[8] T. F. Smith and M. S. Waterman. Identification of common molecular subsequences. *Journal of Molecular Biology*, 147:195–197, 1981.

[9] Naftali Tishby, Fernando C. Pereira, and William Bialek. The information bottleneck method. In *Proc. of the 37-th Annual Allerton Conference on Communication, Control and Computing*, pages 368–77, 1999.

[10] Golan Yona and Michael Levitt. Within the twilight zone: A sensitive profile-profile comparison tool based on information theory. *Journal of Molecular Biology*, 315:1257–75, 2002.
